# Efficient Learning of Sparse Representations with an Energy-Based Model

**Marc'Aurelio Ranzato   Christopher Poultney   Sumit Chopra   Yann LeCun**
Courant Institute of Mathematical Sciences
New York University, New York, NY 10003
{ranzato,crispy,sumit,yann}@cs.nyu.edu

## Abstract

We describe a novel unsupervised method for learning sparse, overcomplete features. The model uses a linear encoder, and a linear decoder preceded by a sparsifying non-linearity that turns a code vector into a quasi-binary sparse code vector. Given an input, the optimal code minimizes the distance between the output of the decoder and the input patch while being as similar as possible to the encoder output. Learning proceeds in a two-phase EM-like fashion: (1) compute the minimum-energy code vector, (2) adjust the parameters of the encoder and decoder so as to decrease the energy. The model produces "stroke detectors" when trained on handwritten numerals, and Gabor-like filters when trained on natural image patches. Inference and learning are very fast, requiring no preprocessing, and no expensive sampling. Using the proposed unsupervised method to initialize the first layer of a convolutional network, we achieved an error rate slightly lower than the best reported result on the MNIST dataset. Finally, an extension of the method is described to learn topographical filter maps.

## 1   Introduction

Unsupervised learning methods are often used to produce pre-processors and feature extractors for image analysis systems. Popular methods such as Wavelet decomposition, PCA, Kernel-PCA, Non-Negative Matrix Factorization [1], and ICA produce compact representations with somewhat uncorrelated (or independent) components [2]. Most methods produce representations that either preserve or reduce the dimensionality of the input. However, several recent works have advocated the use of sparse-overcomplete representations for images, in which the dimension of the feature vector is *larger* than the dimension of the input, but only a small number of components are non-zero for any one image [3, 4]. Sparse-overcomplete representations present several potential advantages. Using high-dimensional representations increases the likelihood that image categories will be easily (possibly linearly) separable. Sparse representations can provide a simple interpretation of the input data in terms of a small number of "parts" by extracting the structure hidden in the data. Furthermore, there is considerable evidence that biological vision uses sparse representations in early visual areas [5, 6].

It seems reasonable to consider a representation "complete" if it is possible to reconstruct the input from it, because the information contained in the input would need to be preserved in the representation itself. Most unsupervised learning methods for feature extraction are based on this principle, and can be understood in terms of an *encoder* module followed by a *decoder* module. The encoder takes the input and computes a code vector, for example a sparse and overcomplete representation. The decoder takes the code vector given by the encoder and produces a reconstruction of the input. Encoder and decoder are trained in such a way that reconstructions provided by the decoder are as similar as possible to the actual input data, when these input data have the same statistics as the training samples. Methods such as Vector Quantization, PCA, auto-encoders [7], Restricted Boltzmann Machines [8], and others [9] have exactly this architecture but with different constraints on the code and learning algorithms, and different kinds of encoder and decoder architectures. In other approaches, the encoding module is missing but its role is taken by a minimization in code

space which retrieves the representation [3]. Likewise, in non-causal models the decoding module is missing and sampling techniques must be used to reconstruct the input from a code [4]. In sec. 2, we describe an *energy-based* model which has both an encoding and a decoding part. After training, the encoder allows very fast inference because finding a representation does not require solving an optimization problem. The decoder provides an easy way to reconstruct input vectors, thus allowing the trainer to assess directly whether the representation extracts most of the information from the input.

Most methods find representations by minimizing an appropriate loss function during training. In order to learn sparse representations, a term enforcing sparsity is added to the loss. This term usually penalizes those code units that are active, aiming to make the distribution of their activities highly peaked at zero with heavy tails [10] [4]. A drawback for these approaches is that some action might need to be taken in order to prevent the system from always activating the same few units and collapsing all the others to zero [3]. An alternative approach is to embed a sparsifying module, e.g. a non-linearity, in the system [11]. This in general forces all the units to have the same degree of sparsity, but it also makes a theoretical analysis of the algorithm more complicated. In this paper, we present a system which achieves sparsity by placing a non-linearity between encoder and decoder. Sec. 2.1 describes this module, dubbed the "*Sparsifying Logistic*", which is a logistic function with an adaptive bias that tracks the mean of its input. This non-linearity is parameterized in a simple way which allows us to control the degree of sparsity of the representation as well as the entropy of each code unit.

Unfortunately, learning the parameters in encoder and decoder can not be achieved by simple back-propagation of the gradients of the reconstruction error: the Sparsifying Logistic is highly non-linear and resets most of the gradients coming from the decoder to zero. Therefore, in sec. 3 we propose to augment the loss function by considering not only the parameters of the system but also the code vectors as variables over which the optimization is performed. Exploiting the fact that 1) it is fairly easy to determine the weights in encoder and decoder when "good" codes are given, and 2) it is straightforward to compute the optimal codes when the parameters in encoder and decoder are fixed, we describe a simple iterative coordinate descent optimization to learn the parameters of the system. The procedure can be seen as a sort of *deterministic version of the EM algorithm* in which the code vectors play the role of hidden variables. The learning algorithm described turns out to be particularly simple, fast and robust. No pre-processing is required for the input images, beyond a simple centering and scaling of the data. In sec. 4 we report experiments of feature extraction on handwritten numerals and natural image patches. When the system has a linear encoder and decoder (remember that the Sparsifying Logistic is a separate module), the filters resemble "object parts" for the numerals, and localized, oriented features for the natural image patches. Applying these features for the classification of the digits in the MNIST dataset, we have achieved by a small margin the best accuracy ever reported in the literature. We conclude by showing a hierarchical extension which suggests the form of simple and complex cell receptive fields, and leads to a topographic layout of the filters which is reminiscent of the topographic maps found in area V1 of the visual cortex.

## 2 The Model

The proposed model is based on three main components, as shown in fig. 1:

- The *encoder*: A set of feed-forward filters parameterized by the rows of matrix $W_C$, that computes a code vector from an image patch $X$.
- The *Sparsifying Logistic*: A non-linear module that transforms the code vector $Z$ into a sparse code vector $\bar{Z}$ with components in the range $[0, 1]$.
- The *decoder*: A set of reverse filters parameterized by the columns of matrix $W_D$, that computes a reconstruction of the input image patch from the sparse code vector $\bar{Z}$.

The *energy* of the system is the sum of two terms:

$$E(X, Z, W_C, W_D) = E_C(X, Z, W_C) + E_D(X, Z, W_D) \tag{1}$$

The first term is the *code prediction energy* which measures the discrepancy between the output of the encoder and the code vector $Z$. In our experiments, it is defined as:

$$E_C(X, Z, W_C) = \frac{1}{2}||Z - \text{Enc}(X, W_C)||^2 = \frac{1}{2}||Z - W_C X||^2 \tag{2}$$

The second term is the *reconstruction energy* which measures the discrepancy between the reconstructed image patch produced by the decoder and the input image patch $X$. In our experiments, it

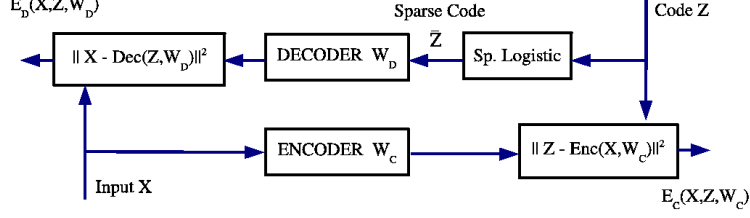

Figure 1: Architecture of the energy-based model for learning sparse-overcomplete representations. The input image patch $X$ is processed by the *encoder* to produce an initial estimate of the code vector. The *encoding prediction energy* $E_C$ measures the squared distance between the code vector $Z$ and its estimate. The code vector $Z$ is passed through the *Sparsifying Logistic* non-linearity which produces a sparsified code vector $\bar{Z}$. The *decoder* reconstructs the input image patch from the sparse code. The *reconstruction energy* $E_D$ measures the squared distance between the reconstruction and the input image patch. The optimal code vector $Z^*$ for a given patch minimizes the sum of the two energies. The learning process finds the encoder and decoder parameters that minimize the energy for the optimal code vectors averaged over a set of training samples.

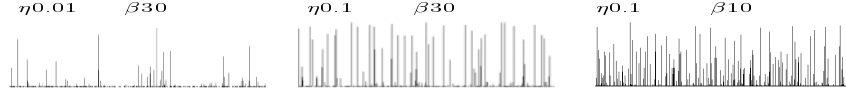

Figure 2: Toy example of sparsifying rectification produced by the *Sparsifying Logistic* for different choices of the parameters $\eta$ and $\beta$. The input is a sequence of Gaussian random variables. The output, computed by using eq. 4, is a sequence of spikes whose rate and amplitude depend on the parameters $\eta$ and $\beta$. In particular, increasing $\beta$ has the effect of making the output approximately binary, while increasing $\eta$ increases the firing rate of the output signal.

is defined as:

$$E_D(X, Z, W_D) = \frac{1}{2}||X - \text{Dec}(\bar{Z}, W_D)||^2 = \frac{1}{2}||X - W_D\bar{Z}||^2 \tag{3}$$

where $\bar{Z}$ is computed by applying the Sparsifying Logistic non-linearity to $Z$.

## 2.1 The Sparsifying Logistic

The *Sparsifying Logistic* module is a non-linear front-end to the decoder that transforms the code vector into a sparse vector with positive components. Let us consider how it transforms the $k$-th training sample. Let $z_i(k)$ be the $i$-th component of the code vector and $\bar{z}_i(k)$ be its corresponding output, with $i \in [1..m]$ where $m$ is the number of components in the code vector. The relation between these variables is given by:

$$\bar{z}_i(k) = \frac{\eta e^{\beta z_i(k)}}{\zeta_i(k)}, i \in [1..m] \quad \text{with} \quad \zeta_i(k) = \eta e^{\beta z_i(k)} + (1-\eta)\zeta_i(k-1) \tag{4}$$

where it is assumed that $\eta \in [0, 1]$. $\zeta_i(k)$ is the weighted sum of values of $e^{\beta z_i(n)}$ corresponding to previous training samples $n$, with $n \leq k$. The weights in this sum are exponentially decaying as can be seen by unrolling the recursive equation in 4. This non-linearity can be easily understood as a weighted softmax function applied over consecutive samples of the same code unit. This produces a sequence of positive values which, for large values of $\beta$ and small values of $\eta$, is characterized by brief and punctuate activities in time. This behavior is reminiscent of the spiking behavior of neurons. $\eta$ controls the sparseness of the code by determining the "width" of the time window over which samples are summed up. $\beta$ controls the degree of "softness" of the function. Large $\beta$ values yield quasi-binary outputs, while small $\beta$ values produce more graded responses; fig. 2 shows how these parameters affect the output when the input is a Gaussian random variable.

Another view of the Sparsifying Logistic is as a logistic function with an adaptive bias that tracks the average input; by dividing the right hand side of eq. 4 by $\eta e^{\beta z_i(k)}$ we have:

$$\bar{z}_i(k) = \left[1 + e^{-\beta(z_i(k) - \frac{1}{\beta}\log(\frac{1-\eta}{\eta}\zeta_i(k-1)))}\right]^{-1}, \quad i \in [1..m] \tag{5}$$

Notice how $\beta$ directly controls the gain of the logistic. Large values of this parameter will turn the non-linearity into a step function and will make $\bar{Z}(k)$ a binary code vector.

In our experiments, $\zeta_i$ is treated as trainable parameter and kept fixed after the learning phase. In this case, the Sparsifying Logistic reduces to a logistic function with a fixed gain and a learned bias. For large $\beta$ in the continuous-time limit, the spikes can be shown to follow a homogeneous Poisson process. In this framework, sparsity is a "temporal" property characterizing each single unit in the code, rather than a "spatial" property shared among all the units in a code. Spatial sparsity usually requires some sort of ad-hoc normalization to ensure that the components of the code that are "on" are not always the same ones. Our solution tackles this problem differently: each unit must be sparse when encoding different samples, independently from the activities of the other components in the code vector. Unlike other methods [10], no ad-hoc rescaling of the weights or code units is necessary.

## 3   Learning

Learning is accomplished by minimizing the energy in eq. 1. Indicating with superscripts the indices referring to the training samples and making explicit the dependencies on the code vectors, we can rewrite the energy of the system as:

$$E(W_C, W_D, Z^1, \ldots, Z^P) = \sum_{i=1}^{P} [E_D(X^i, Z^i, W_D) + E_C(X^i, Z^i, W_C)] \tag{6}$$

This is also the loss function we propose to minimize during training. The parameters of the system, $W_C$ and $W_D$, are found by solving the following minimization problem:

$$\{W_C^*, W_D^*\} = argmin_{\{W_c, W_d\}} min_{Z^1, \ldots, Z^P} E(W_c, W_d, Z^1, \ldots, Z^P) \tag{7}$$

It is easy to minimize this loss with respect to $W_C$ and $W_D$ when the $Z^i$ are known and, particularly for our experiments where encoder and decoder are a set of linear filters, this is a convex quadratic optimization problem. Likewise, when the parameters in the system are fixed it is straightforward to minimize with respect to the codes $Z^i$. These observations suggest a coordinate descent optimization procedure. First, we find the optimal $Z^i$ for a given set of filters in encoder and decoder. Then, we update the weights in the system fixing $Z^i$ to the value found at the previous step. We iterate these two steps in alternation until convergence. In our experiments we used an *on-line* version of this algorithm which can be summarized as follows:

1. propagate the input $X$ through the encoder to get a codeword $Z_{init}$
2. minimize the loss in eq. 6, sum of reconstruction and code prediction energy, with respect to $Z$ by gradient descent using $Z_{init}$ as the initial value
3. compute the gradient of the loss with respect to $W_C$ and $W_D$, and perform a gradient step

where the superscripts have been dropped because we are referring to a generic training sample. Since the code vector $Z$ minimizes both energy terms, it not only minimizes the reconstruction energy, but is also as similar as possible to the code predicted by the encoder. After training the decoder settles on filters that produce low reconstruction errors from minimum-energy, sparsified code vectors $\bar{Z}^*$, while the encoder simultaneously learns filters that predict the corresponding minimum-energy codes $Z^*$. In other words, the system converges to a state where minimum-energy code vectors not only reconstruct the image patch but can also be easily predicted by the encoder filters. Moreover, starting the minimization over $Z$ from the prediction given by the encoder allows convergence in very few iterations. After the first few thousand training samples, the minimization over $Z$ requires just 4 iterations on average. When training is complete, a simple pass through the encoder will produce an accurate prediction of the minimum-energy code vector. In the experiments, two regularization terms are added to the loss in eq. 6: a "lasso" term equal to the $L_1$ norm of $W_C$ and $W_D$, and a "ridge" term equal to their $L_2$ norm. These have been added to encourage the filters to localize and to suppress noise.

Notice that we could differently weight the encoding and the reconstruction energies in the loss function. In particular, assigning a very large weight to the encoding energy corresponds to turning the penalty on the encoding prediction into a *hard* constraint. The code vector would be assigned the value predicted by the encoder, and the minimization would reduce to a mean square error minimization through back-propagation as in a standard autoencoder. Unfortunately, this autoencoder-like

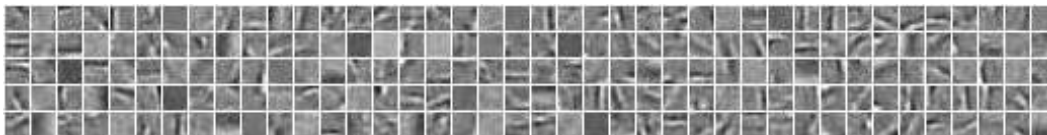

Figure 3: Results of feature extraction from 12x12 patches taken from the Berkeley dataset, showing the 200 filters learned by the decoder.

learning fails because Sparsifying Logistic is almost always highly saturated (otherwise the representation would not be sparse). Hence, the gradients back-propagated to the encoder are likely to be very small. This causes the direct minimization over encoder parameters to fail, but does not seem to adversely affect the minimization over code vectors. We surmise that the large number of degrees of freedom in code vectors (relative to the number of encoder parameters) makes the minimization problem considerably better conditioned. In other words, the alternated descent algorithm performs a minimization over a much larger set of variables than regular back-prop, and hence is less likely to fall victim to local minima. The alternated descent over code and parameters can be seen as a kind of *deterministic EM*. It is related to gradient-descent over parameters (standard back-prop) in the same way that the EM algorithm is related to gradient ascent for maximum likelihood estimation.

This learning algorithm is not only simple but also very fast. For example, in the experiments of sec. 4.1 it takes less than 30 minutes on a 2GHz processor to learn 200 filters from 100,000 patches of size 12x12, and after just a few minutes the filters are already very similar to the final ones. This is much more efficient and robust than what can be achieved using other methods. For example, in Olshausen and Field's [10] linear generative model, inference is expensive because minimization in code space is necessary during testing as well as training. In Teh et al. [4], learning is very expensive because the decoder is missing, and sampling techniques [8] must be used to provide a reconstruction. Moreover, most methods rely on pre-processing of the input patches such as whitening, PCA and low-pass filtering in order to improve results and speed up convergence. In our experiments, we need only center the data by subtracting a global mean and scale by a constant.

# 4   Experiments

In this section we present some applications of the proposed energy-based model. Two standard data sets were used: natural image patches and handwritten digits. As described in sec. 2, the encoder and decoder learn linear filters. As mentioned in sec. 3, the input images were only trivially pre-processed.

## 4.1   Feature Extraction from Natural Image Patches

In the first experiment, the system was trained on 100,000 gray-level patches of size 12x12 extracted from the Berkeley segmentation data set [12]. Pre-processing of images consists of subtracting the global mean pixel value (which is about 100), and dividing the result by 125. We chose an overcomplete factor approximately equal to 2 by representing the input with 200 code units[1]. The Sparsifying Logistic parameters $\eta$ and $\beta$ were equal to 0.02 and 1, respectively. The learning rate for updating $W_C$ was set to 0.005 and for $W_D$ to 0.001. These are decreased progressively during training. The coefficients of the $L_1$ and $L_2$ regularization terms were about 0.001. The learning rate for the minimization in code space was set to 0.1, and was multiplied by 0.8 every 10 iterations, for at most 100 iterations. Some components of the sparse code must be allowed to take continuous values to account for the average value of a patch. For this reason, during training we saturated the running sums $\zeta$ to allow some units to be always active. Values of $\zeta$ were saturated to $10^9$. We verified empirically that subtracting the local mean from each patch eliminates the need for this saturation. However, saturation during training makes testing less expensive. Training on this data set takes less than half an hour on a 2GHz processor.

Examples of learned encoder and decoder filters are shown in figure 3. They are spatially localized, and have different orientations, frequencies and scales. They are somewhat similar to, but more localized than, Gabor wavelets and are reminiscent of the receptive fields of V1 neurons. Interest-

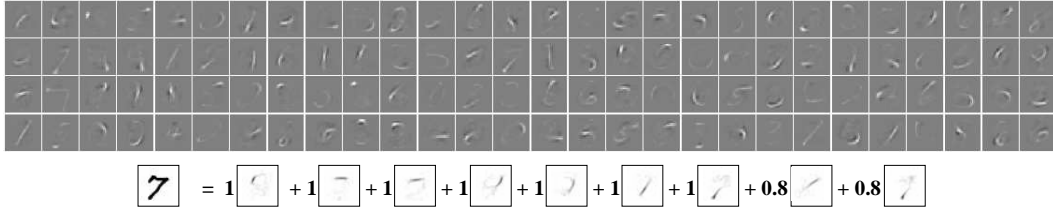

Figure 4: Top: A randomly selected subset of encoder filters learned by our energy-based model when trained on the MNIST handwritten digit dataset. Bottom: An example of reconstruction of a digit randomly extracted from the test data set. The reconstruction is made by adding "parts": it is the *additive* linear combination of few basis functions of the decoder with positive coefficients.

ingly, the encoder and decoder filter values are nearly identical up to a scale factor. After training, inference is extremely fast, requiring only a simple matrix-vector multiplication.

## 4.2 Feature Extraction from Handwritten Numerals

The energy-based model was trained on 60,000 handwritten digits from the MNIST data set [13], which contains quasi-binary images of size 28x28 (784 pixels). The model is the same as in the previous experiment. The number of components in the code vector was 196. While 196 is less than the 784 inputs, the representation is still overcomplete, because the effective dimension of the digit dataset is considerably less than 784. Pre-processing consisted of dividing each pixel value by 255. Parameters $\eta$ and $\beta$ in the temporal softmax were 0.01 and 1, respectively. The other parameters of the system have been set to values similar to those of the previous experiment on natural image patches. Each one of the filters, shown in the top part of fig. 4, contains an elementary "part" of a digit. Straight stroke detectors are present, as in the previous experiment, but curly strokes can also be found. Reconstruction of most single digits can be achieved by a linear additive combination of a small number of filters since the output of the Sparsifying Logistic is sparse and positive. The bottom part of fig. 4 illustrates this reconstruction by "parts".

## 4.3 Learning Local Features for the MNIST dataset

Deep convolutional networks trained with backpropagation hold the current record for accuracy on the MNIST dataset [14, 15]. While back-propagation produces good low-level features, it is well known that deep networks are particularly challenging for gradient-descent learning. Hinton et al. [16] have recently shown that initializing the weights of a deep network using unsupervised learning before performing supervised learning with back-propagation can significantly improve the performance of a deep network. This section describes a similar experiment in which we used the proposed method to initialize the first layer of a large convolutional network. We used an architecture essentially identical to *LeNet-5* as described in [15]. However, because our model produces sparse features, our network had a considerably larger number of feature maps: 50 for layer 1 and 2, 50 for layer 3 and 4, 200 for layer 5, and 10 for the output layer. The numbers for LeNet-5 were 6, 16, 100, and 10 respectively. We refer to our larger network as the 50-50-200-10 network. We trained this networks on 55,000 samples from MNIST, keeping the remaining 5,000 training samples as a validation set. When the error on the validation set reached its minimum, an additional five sweeps were performed on the training set augmented with the validation set (unless this increased the training loss). Then the learning was stopped, and the final error rate on the test set was measured. When the weights are initialized randomly, the 50-50-200-10 achieves a test error rate of 0.7%, to be compared with the 0.95% obtained by [15] with the 6-16-100-10 network.

In the next experiment, the proposed sparse feature learning method was trained on 5x5 image patches extracted from the MNIST training set. The model had a 50-dimensional code. The encoder filters were used to initialize the first layer of the 50-50-200-10 net. The network was then trained in the usual way, except that the first layer was kept fixed for the first 10 epochs through the training set. The 50 filters after training are shown in fig. 5. The test error rate was 0.6%. To our knowledge, this is the best results ever reported with a method trained on the original MNIST set, without deskewing nor augmenting the training set with distorted samples.

The training set was then augmented with samples obtained by elastically distorting the original training samples, using a method similar to [14]. The error rate of the 50-50-200-10 net with random initialization was 0.49% (to be compared to 0.40% reported in [14]). By initializing the first layer

with the filters obtained with the proposed method, the test error rate dropped to 0.39%. While this is the best numerical result ever reported on MNIST, it is not statistically different from [14].

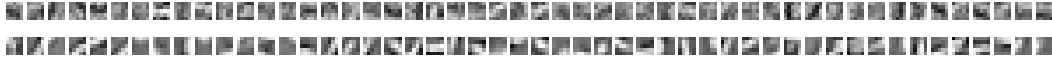

Figure 5: Filters in the first convolutional layer after training when the network is randomly initialized (top row) and when the first layer of the network is initialized with the features learned by the unsupervised energy-based model (bottom row).

| Architecture | Training Set Size | | | | | |
|---|---|---|---|---|---|---|
| | 20K | | 60K | | 60K + Distortions | |
| 6-16-100-10 [15] | - | - | 0.95 | - | 0.60 | - |
| 5-50-100-10 [14] | - | - | - | - | 0.40 | - |
| 50-50-200-10 | 1.01 | **0.89** | 0.70 | **0.60** | 0.49 | **0.39** |

Table 1: Comparison of test error rates on MNIST dataset using convolutional networks with various training set size: 20,000, 60,000, and 60,000 plus 550,000 elastic distortions. For each size, results are reported with randomly initialized filters, and with first-layer filters initialized using the proposed algorithm (bold face).

## 4.4 Hierarchical Extension: Learning Topographic Maps

It has already been observed that features extracted from natural image patches resemble Gabor-like filters, see fig. 3. It has been recently pointed out [6] that these filters produce codes with somewhat uncorrelated but not independent components. In order to capture higher order dependencies among code units, we propose to extend the encoder architecture by adding to the linear filter bank a second layer of units. In this hierarchical model of the encoder, the units produced by the filter bank are now laid out on a two dimensional grid and filtered according to a fixed weighted mean kernel. This assigns a larger weight to the central unit and a smaller weight to the units in the periphery. In order to activate a unit at the output of the Sparsifying Logistic, all the afferent unrectified units in the first layer must agree in giving a strong positive response to the input patch. As a consequence neighboring filters will exhibit similar features. Also, the top level units will encode features that are more translation and rotation invariant, *de facto* modeling complex cells. Using a neighborhood of size 3x3, toroidal boundary conditions, and computing code vectors with 400 units from 12x12 input patches from the Berkeley dataset, we have obtained the topographic map shown in fig. 6. Filters exhibit features that are locally similar in orientation, position, and phase. There are two low frequency clusters and pinwheel regions similar to what is experimentally found in cortical topography.

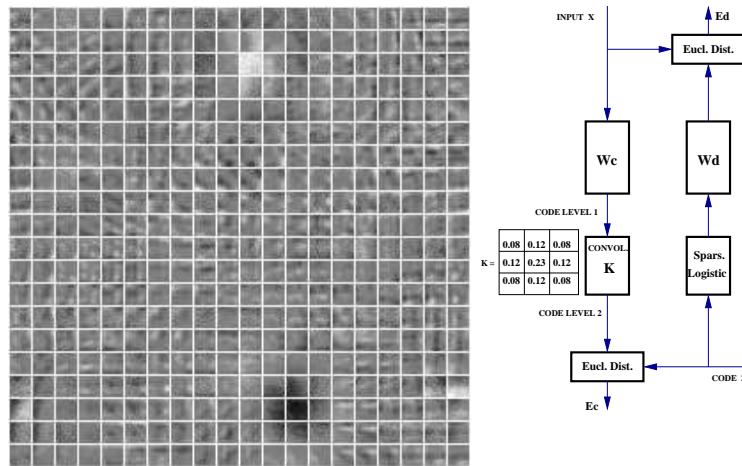

Figure 6: Example of filter maps learned by the topographic hierarchical extension of the model. The outline of the model is shown on the right.

# 5 Conclusions

An energy-based model was proposed for unsupervised learning of sparse overcomplete representations. Learning to extract sparse features from data has applications in classification, compression, denoising, inpainting, segmentation, and super-resolution interpolation. The model has none of the inefficiencies and idiosyncrasies of previously proposed sparse-overcomplete feature learning methods. The decoder produces accurate reconstructions of the patches, while the encoder provides a fast prediction of the code without the need for any particular preprocessing of the input images.

It seems that a non-linearity that directly sparsifies the code is considerably simpler to control than adding a sparsity term in the loss function, which generally requires ad-hoc normalization procedures [3].

In the current work, we used linear encoders and decoders for simplicity, but the model authorizes non-linear modules, as long as gradients can be computed and back-propagated through them. As briefly presented in sec. 4.4, it is straightforward to extend the original framework to hierarchical architectures in encoder, and the same is possible in the decoder. Another possible extension would stack multiple instances of the system described in the paper, with each system as a module in a multi-layer structure where the sparse code produced by one feature extractor is fed to the input of a higher-level feature extractor.

Future work will include the application of the model to various tasks, including facial feature extraction, image denoising, image compression, inpainting, classification, and invariant feature extraction for robotics applications.

### Acknowledgments

We wish to thank Sebastian Seung and Geoff Hinton for helpful discussions. This work was supported in part by the NSF under grants No. 0325463 and 0535166, and by DARPA under the LAGR program.

## Footnotes

[1]Overcompleteness must be evaluated by considering the number of code units and the effective dimensionality of the input as given by PCA.

# References

[1] Lee, D.D. and Seung, H.S. (1999) Learning the parts of objects by non-negative matrix factorization. Nature, 401:788-791.

[2] Hyvarinen, A. and Hoyer, P.O. (2001) A 2-layer sparse coding model learns simple and complex cell receptive fields and topography from natural images. Vision Research, 41:2413-2423.

[3] Olshausen, B.A. (2002) Sparse codes and spikes. R.P.N. Rao, B.A. Olshausen and M.S. Lewicki Eds. - MIT press:257-272.

[4] Teh, Y.W. and Welling, M. and Osindero, S. and Hinton, G.E. (2003) Energy-based models for sparse overcomplete representations. Journal of Machine Learning Research, 4:1235-1260.

[5] Lennie, P. (2003) The cost of cortical computation. Current biology, 13:493-497

[6] Simoncelli, E.P. (2005) Statistical modeling of photographic images. Academic Press 2nd ed.

[7] Hinton, G.E. and Zemel, R.S. (1994) Autoencoders, minimum description length, and Helmholtz free energy. Advances in Neural Information Processing Systems 6, J. D. Cowan, G. Tesauro and J. Alspector (Eds.), Morgan Kaufmann: San Mateo, CA.

[8] Hinton, G.E. (2002) Training products of experts by minimizing contrastive divergence. Neural Computation, 14:1771-1800.

[9] Doi E., Balcan, D.C. and Lewicki, M.S. (2006) A theoretical analysis of robust coding over noisy overcomplete channels. Advances in Neural Information Processing Systems 18, MIT Press.

[10] Olshausen, B.A. and Field, D.J. (1997) Sparse coding with an overcomplete basis set: a strategy employed by V1? Vision Research, 37:3311-3325.

[11] Foldiak, P. (1990) Forming sparse representations by local anti-hebbian learning. Biological Cybernetics, 64:165-170.

[12] The berkeley segmentation dataset http://www.cs.berkeley.edu/projects/vision/grouping/segbench/

[13] The MNIST database of handwritten digits http://yann.lecun.com/exdb/mnist/

[14] Simard, P.Y. Steinkraus, D. and Platt, J.C. (2003) Best practices for convolutional neural networks. ICDAR

[15] LeCun, Y. Bottou, L. Bengio, Y. and Haffner, P. (1998) Gradient-based learning applied to document recognition. Proceedings of the IEEE, 86(11):2278-2324.

[16] Hinton, G.E., Osindero, S. and Teh, Y. (2006) A fast learning algorithm for deep belief nets. Neural Computation 18, pp 1527-1554.
